# Perceiving Complex Visual Scenes: An Oscillator Neural Network Model that Integrates Selective Attention, Perceptual Organisation, and Invariant Recognition

**Rainer Goebel**
Department of Psychology
University of Braunschweig
Spielmannstr. 19
W-3300 Braunschweig, Germany

## Abstract

Which processes underly our ability to quickly recognize familiar objects within a complex visual input scene? In this paper an implemented neural network model is described that attempts to specify how selective visual attention, perceptual organisation, and invariance transformations might work together in order to segment, select, and recognize objects out of complex input scenes containing multiple, possibly overlapping objects. Retinotopically organized feature maps serve as input for two main processing routes: the 'where-pathway' dealing with location information and the 'what-pathway' computing the shape and attributes of objects. A location-based attention mechanism operates on an early stage of visual processing selecting a contigous region of the visual field for preferential processing. Additionally, location-based attention plays an important role for invariant object recognition controling appropriate normalization processes within the what-pathway. Object recognition is supported through the segmentation of the visual field into distinct entities. In order to represent different segmented entities at the same time, the model uses an oscillatory binding mechanism. Connections between the where-pathway and the what-pathway lead to a flexible cooperation between different functional subsystems producing an overall behavior which is consistent with a variety of psychophysical data.

# 1   INTRODUCTION

We are able to recognize a familiar object from many different viewpoints. Additionally, an object normally does not appear in isolation but in combination with other objects. These varying viewing conditions produce very different retinal neural representations. The task of the visual system can be considered as a transformation process forming high-level object representations which are invariant with respect to different viewing conditions. Selective attention and perceptual organisation seem to play an important role in this transformation process.

## 1.1   LOCATION-BASED VS OBJECT-BASED ATTENTION

Neisser (1967) assumed that visual processing is done in two stages: an early stage that operates in parallel across the entire visual field, and a later stage that can only process information from a limited part of the field at any one time. Neisser (1967) proposed an *object-based approach* to selective attention: the first, 'preattentive', stage segments the whole field into seperate objects on the basis of Gestalt principles; the second stage, focal attention, *selects one of these objects* for detailed analysis.
Other theories stress the *location-based* nature of visual attention: a limited contigous *region* is filtered for detailed analysis (e.g., Posner et al., 1980). There exists a number of models of location-based attention (e.g., Hinton & Lang, 1985; Mozer, 1991; Sandon, 1990) and a few models of object-based attention using whole object knowledge (e.g., Fukushima, 1986). Our model attempts to integrate both approaches: location-based attention - implemented as a 'spotlight' - operates on an early stage of visual processing selecting a contigous region for detailed processing. However, the position and the size of the attentional window is determined to a large extent from the results of a segmentation process operating at different levels within the system.

## 1.2   DYNAMIC BINDING

The question of how groupings can be represented in a neural network is known as the binding problem. It occurs in many variations, e.g., as the problem of how to represent multiple objects simultaneously but sufficiently distinct that confusions ('illusory conjunctions') at later processing stages are avoided.
An interesting solution of the binding problem is based on ideas proposed by Milner (1974) and von der Malsburg (1981). In contrast to most connectionist models assuming that only the *average output activity* of neurons encodes important information, they suggest that the *exact timing* of neuronal activity (the firing of individual neurons or the 'bursting' of cell groups) plays an important role for information processing in the brain. The central idea is that stimulated units do not respond with a constant output but with *oscillatory behavior* which can be exploited to represent feature linkings. A possible solution for representing multiple objects might be that the parts of one object are bound together through synchronized (phase-locked) oscillations and separated from other objects through an uncorellated phase relation. Recent empirical findings (Eckhorn et al., 1988; Gray & Singer, 1989) provide some evidence that the brain may indeed use phase-locked oscillations as a means for representing global object properties.

## 2   THE MODEL

### 2.1   SYSTEM DYNAMICS

In order to establish dynamic binding via phase-locked oscillations the units of the model must be able to exhibit oscillatory behavior. Stimulated from the empirical findings mentioned earlier, a rapidly growing number of work has studied populations of oscillating units (e.g., Eckhorn et al., 1990; Sompolinsky et al., 1990). There exists also a number of models using phase-locked oscillations in order to simulate various aspects of perceptual organisation (e.g., Schillen & König, 1991; Mozer, Zemel, Behrmann & Williams, 1992). We defined computationally simple model neurons which allow to represent independently an activation value and a period value. Such a model neuron possesses two types of input areas: the *activation gate* (a-gate) and the *period-gate* (p-gate) which allow the model neurons to communicate via two types of connections (cf. Eckhorn et al., 1990; they distinguish between 'feeding' and 'linking' connections). We make the following definitions:

- $w_{ij}^a$: weight from model neuron $j$ to the a-gate of model neuron $i$.
- $w_{ij}^p$: weight from model neuron $j$ to the p-gate of model neuron $i$.
- $\xi_i(t)$: internal time-keeper of unit $i$
- $T$: globally defined period length
- $T_i(N)$: period length of unit $i$ (Nth oscillation)

Each model neuron possesses an internal time-keeper $\xi_i(t)$ counting the number of bins elapsed since the last firing point. A model neuron is refractory until the time-keeper reaches the value $T_i$ (e.g., $T_i = T = 8$). Then it may emit an activation value and resets the time-keeper. Depending on the stimulation received at the p-gate (see below) a model neuron fires either if $\xi = T - 1$ or $\xi = T$. This variation of the individual period length $T_i$ is the only possibility for a unit to change its phase relation to other units. The value of the globally defined period length $T$ determines directly how many objects may be represented 'simultaneously'.

The activation value $a_i$ at the internal time $\xi$ is determined as follows:

$$net_i(\xi = T_i) = \sum_{\xi=1}^{T_i} \sum_{j=1}^{n} w_{ij}^a a_j(\xi) \tag{1}$$

$$a_i(\xi) = \begin{cases} (1 - \tau)a_i(\xi - T_i) + \tau\,\sigma(net_i(\xi) + b_i) & \text{if } \xi = T_i \\ 0 & \text{otherwise} \end{cases} \tag{2}$$

where $\sigma(x)$ is the logistic (sigmoidal) function. If we consider an extreme case with $T = 1$ we obtain the following equations:

$$net_i(t) = \sum_{j=1}^{n} w_{ij}^a a_j \tag{3}$$

$$a_i(t) = (1 - \tau)a_i(t-1) + \tau\,\sigma(net_i(t) + b_i) \tag{4}$$

This derivation allows us to study the same network as a conventional connectionist network ($T = 1$) with a 'non-oscillatory' activation function to which we can add a dynamic binding mechanism by simply setting $T > 1$. In the latter case the input at the p-gate determines the length of the current period as either $T_i = T - 1$ or $T_i = T$. The decision to shift the phase relation to other neurons should be done in such a way that the 'belongingness constraints' imposed by the connectivity pattern of the p-weights $w_{ij}^p$ is maximized, e.g., if two units are positively p-coupled they should oscillate in phase, if they are negatively p-coupled they should oscillate out of phase. The decision whether a unit fires at $T - 1$ or $T$ depends on two values, the stimulation received during the refractory period $1 \le \xi < T_i(N - 1)$ and on the stimulation received at the last firing point $\xi = T_i(N - 1)$. These values behave as two opposite forces $g_i$ determining the probability $P_i^<$ of shortening the next period:

$$g_i^1 \quad\quad = \sum_{j=1}^n w_{ij}^p a_j(\xi) \quad\quad\quad \text{if } \xi = T_i \tag{5}$$

$$g_i^2 \quad = \sum_{\xi=1}^{T_i-1} \sum_{j=1}^n \frac{1}{T_i-\xi+2} w_{ij}^p a_j(\xi) \quad \text{if } 1 \le \xi < T_i \tag{6}$$

$$P_i^< = r + \frac{1 - 2r}{1 + e^{(g_i^1 - g_i^2)}} \tag{7}$$

If the value of $g_i^1 - g_i^2$ is large (e.g., there are many positively p-coupled units firing at the same time) it is unlikely that the unit shortens its next period length. If instead the value of $g_i^2 - g_i^1$ is large (e.g., there are many positively coupled neurons firing just before the considered unit) it is likely that the unit will shorten its next period. There exists also a small overall noise level $r = 0.01$ which allows for symmetry breaking (e.g., if two strongly negatively coupled neurons are accidentally phase-locked).

## 2.2   THE INPUT MODULE

Figure 1 shows an overview of the architecture of the model, called HOTSPOT. An input is presented to the model by clamping on units at the model-retina consisting of two layers with 15x25 units. Each layer is meant to correspond to a different color-sensitive ganglion cell type. The retinal representation is then analyzed within different retinotopically organized feature maps (4 oriented line segments and 2 unoriented color blobs) as a simplified representation of an early visual processing stage (corresponding roughly to V1). A lateral connectivity pattern of p-weights within and between these feature maps computes initial feature linkings consistent with the findings of Eckhorn et al., (1988) and Gray and Singer (1989). Each feature map also projects to a second feature-specific layer. The weights between those layers compute the saliency at each position of a particular feature type. These saliency values are finally integrated within the *saliency map*. The retinotopic feature maps project to both the *what pathway*, corresponding roughly to the occipito-temporal processing stream and the *where-pathway*, corresponding to the occipito-parietal stream (e.g., Ungerleider & Mishkin, 1982).

## 2.3 THE SPOTLIGHT-LAYER

The *spotlight-layer* receives bottom-up input from the feature maps via the saliency map and top-down input from the *spotlight-control module*. Based on these sources of stimulation, the spotlight layer computes a circular region of activity representing the current focus of spatial attention. The spotlight-layer corresponds roughly to the pulvinar nucleus of the thalamus. The spotlight-layer gates the flow of information within the what-pathway.

## 2.4 THE WHAT-PATHWAY: FROM FEATURES TO OBJECTS

Processing within the what-pathway includes spatial selection, invariance transformation, complex grouping, object-based selection and object recognition.

### 2.4.1 The Invariance Module

The task of the Invariance module is to retain the spatial arrangement of the features falling within the attentional spotlight while abstracting at the same time the *absolute retinal position* of the attended information. This goal is achieved in several stages along the what-pathway for each feature type. The basic idea is that each neuron connects to several neurons at the next layer. If a certain position is not attended its 'standard' way may be 'open'. If, however, a position is attended, the decision which way is currently gated for a neuron depends on the position and width of the attentional spotlight. Special control layers compute explicitly whether a certain absolute position falls within one of 5 horizontal and 5 vertical regions of the spotlight (e.g., the horizontal regions are 'far left', 'near left', 'center', 'near right', 'far right'). These layers gate the feedforward-synapses within the what-pathway. Finally, the selected information reaches the *invariance-output layers* which have a 7x7 resolution for each feature type. Recently Olshausen, Anderson and Van Essen (1992) proposed a strikingly similar approach for forming invariant representations.

Despite invariance transformations the representation of an object at the invariance-output layers may not be exactly the same as in previous experiences. Therefore the model uses additional processes contributing to invariant object recognition, most importantly the extraction of global features and the exploitation of population codes for the length, position and orientation of features. This also establishes a limited kind of rotation invariance. The selection of information within the what-pathway is consistent with findings from Moran & Desimone (1985): unattended information is excluded from further processing only, if it would stimulate the same population of neurons at the next stage as the selected information.

### 2.4.2 The Object-Recognition-Module

The output of the Invariance Module, the *perceptual-code stage*, feeds to the *object-recognition layer* and receives recurrent connections from that layer terminating both on the a-gate and the p-gate of its units. These connections are trained using the back-propagation learning rule ($T = \tau = 1$). The recurrent loop establishes an interactive recognition process allowing to recognize distorted patterns through the completion of missing information and the suppression of noise.

At the perceptual-code stage perceptual organisation continues based on the initial feature linkings computed within the elementary feature maps. The p-weight pattern

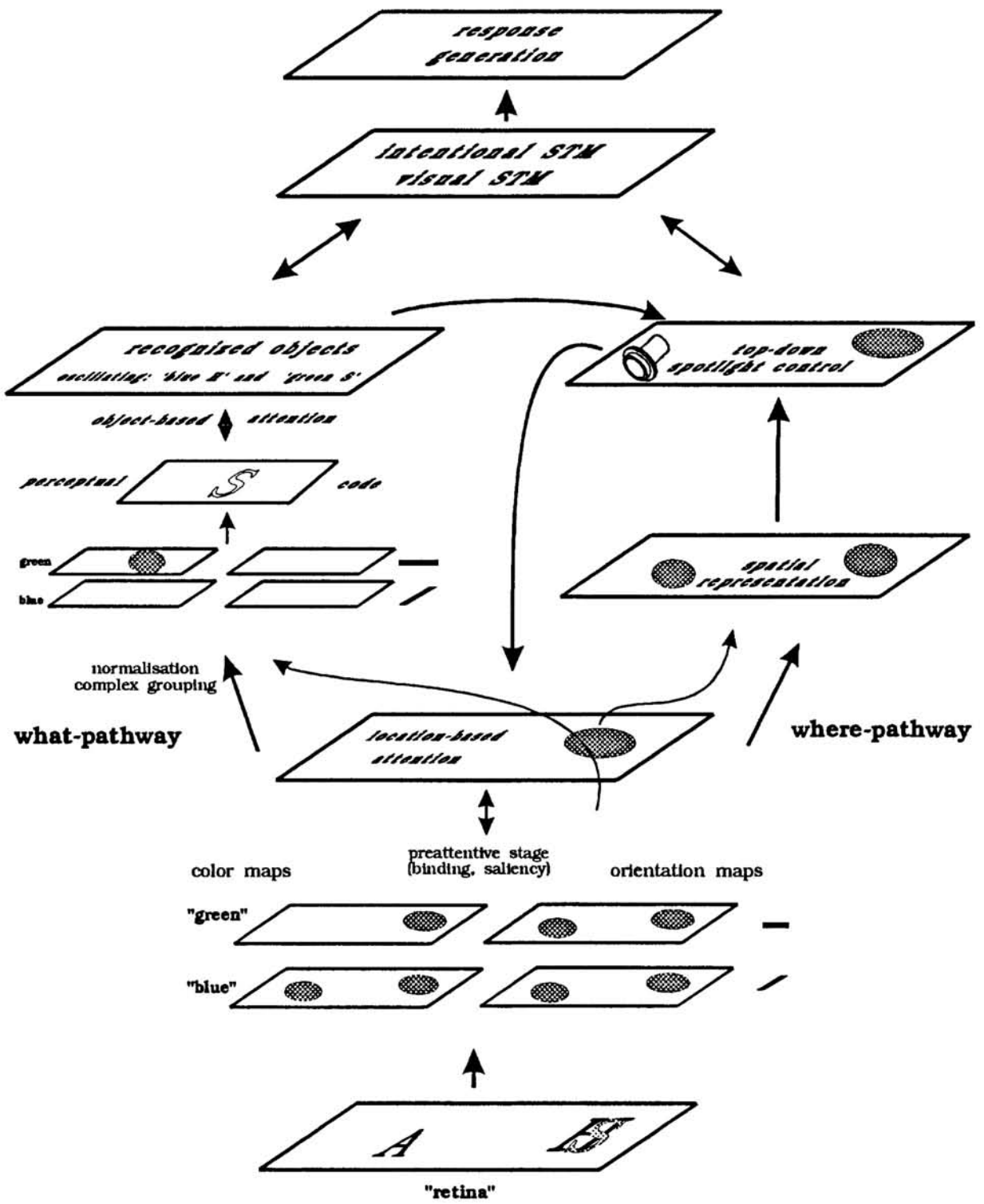

Figure 1: The architecture of HOTSPOT

within the perceptual-code stage implements a set of Gestalt principles such as spatial proximity, similarity and continuity of contour. In additon, acquired shape knowledge is another force acting on the perceptual-code stage in order to bind or separate global features. *Object-based attention* may select one of multiple oscillating objects. For determining a specific object it may use whole-object knowledge (e.g., 'select the letter H'), spatial cues (e.g., 'select the right object') or color cues (e.g., 'select the green object') as well as a combined cue. If the selected object does not use the whole resolution of the perceptual-code stage, commands are sent to the where-pathway in order to adjust the spotlight accordingly.

## 2.5    THE WHERE-PATHWAY

The where-pathway consists of the *saliency map*, the *spotlight-control module*, the *disengagement layer* and the *spatial-representation layer*. The spotlight-control module performs relative movements and size changes of the attentional spotlight which are demanded by the saliency map, object-based selection or commands from a short-term store holding task instructions. If the current position of the spotlight is not changed for some time, the disengagement layer inhibits the corresponding position at the saliency map. The *spatial-representation layer* contains a coarsely tuned representation of all active retinal positions. If no position within the visual field is particularly salient, this layer determines possible target positions for spatial attention.

If the model knows "where what is" this knowledge is transferred to the *visual short-term memory* where a sequence of 'location-object couplings' can be stored.

## 3    CONCLUSION

In this paper an oscillator neural network model was presented that integrates location-based attention, perceptual organisation, and invariance transformations. It was outlined how the cooperation between these mechanisms allow the model to segment, select and recognize objects within a complex input scene. The model was successfully applied to simulate a wide variety of psychophysical data including texture segregation, visual search, hierarchical segmentation and recognition. A typical 'processing cycle' of the model consists of an initial segmentation of the visual field with a broadly tuned spotlight. Then a segmented, but not necessarily recognizable, entity may be selected due to its saliency or by object-based attention. This selection in turn induces movements of the location-based attention mechanism until the selected entity is surrounded by the spotlight. Since in this case appropriate invariance transformations are computed the selected object is optimally recognized. Some predictions of the model concerning the object-based nature of selective attention are currently experimentally tested. HOTSPOT indicates a promising way towards a deeper understanding of complex visual processing by bringing together both neurobiological and psychophysical findings in a fruitful way.

### Acknowledgements

I am grateful to Reinhard Eckhorn, Peter König, Michael Mozer, Werner X. Schneider, Wolf Singer and Dirk Vorberg for valuable discussions.

## References

Eckhorn, R, Bauer, R, Jordan, W., Brosch, M., Kruse, W., Munk, M. & Reitboeck, H.J. (1988) Coherent Oscillations: A mechanism of feature linking in the visual cortex? *Biological Cybernetics*, **60**, 121-130

Eckhorn, R., Reitboeck, H. J., Arndt, M., & Dicke, P. (1990). Feature linking via synchronization among distributed assemblies: The simulation of results from cat visual cortex. *Neural Computation, 2*, 293-307.

Fukushima, K. (1986). A neural network model for selective attention in visual pattern recognition. *Biological Cybernetics, 55*, 5-15.

Gray, M. C. & Singer, W. (1989). Stimulus-specific neuronal oscillations in orientation columns of cat visual cortex. *PNAS USA*, **86**, 1698-1702.

Hinton, G.E, Lang, K.J. (1985). Shape Recognition and Illusory Conjunctions. Proceedings of the 9th IJCAI - Los Angeles, 1, 252-259.

Milner, P.M. (1974). A model for visual shape recognition. *Psych. Rev.*, **81**, 521-535.

Moran, J. & Desimone, R. (1985). Selective attention gates visual processing in the extrastriate cortex. *Science*, **229**, 782-784.

Mozer, M. C. (1991). *The perception of multiple objects: a connectionist approach.* MIT Press / Bradford Books.

Mozer, M. C., Zemel, R. S., & Behrmann, M., Williams, C.K.I. (1992). Learning to segment images using dynamic feature binding. *Neural Computation*, **4**, 650-665.

Neisser, U. (1967). *Cognitive Psychology.* New York: Appleton-Century-Crofts.

Olshausen, B., Anderson, Ch., & Van Essen, D. (1992), A neural model of visual attention and invariant pattern recognition. CNS Memo 18, CalTech.

Posner, M. I., Snyder, C. R. R., & Davidson, B.J. (1980). Attention and the detection of signals. *Journal of Experimental Psychology: General, 109*, 160-174.

Sandon, P. (1990). Simulating visual attention. *Journal of Cog. Neurosc.*, 2, 213-231.

Schillen, Th. B. & König, P. (1991). Stimulus-dependent assembly formation of oscillatory responses: II. Desynchronization. *Neural Computation, 3*, 167-178.

Sompolinsky, H., Golomb, D., & Kleinfeld, D. (1990). Global processing of visual stimuli in a neural network of coupled oscillators. *Proc. Natl. Acad. Sci. USA*, **87**, 7200-7204.

Ungerleider, L. G., & Mishkin, M. (1982). Two cortical visual systems. In D. J. Ingle, M. A. Goodale, & R. J. W. Mansfield (Eds.), *Analysis of visual behavior.* Cambridge, MA: MIT Press.

Van Essen (1985). Functional organization of primate visual cortex. In A. Peters & E. G. Jones (Eds.)., *Cerebral cortex, vol. 3.* New York: Plenum Press.

Von der Malsburg, C. (1981) The correlation theory of brain function. Internal Report 81-2, Dept. of Neurobiology, MPI for Biophysical Chemistry.
